# Isotonic Conditional Random Fields and Local Sentiment Flow

**Yi Mao**
School of Elec. and Computer Engineering
Purdue University - West Lafayette, IN
ymao@ecn.purdue.edu

**Guy Lebanon**
Department of Statistics, and
School of Elec. and Computer Engineering
Purdue University - West Lafayette, IN
lebanon@stat.purdue.edu

## Abstract

We examine the problem of predicting local sentiment flow in documents, and its application to several areas of text analysis. Formally, the problem is stated as predicting an ordinal sequence based on a sequence of word sets. In the spirit of isotonic regression, we develop a variant of conditional random fields that is well-suited to handle this problem. Using the Möbius transform, we express the model as a simple convex optimization problem. Experiments demonstrate the model and its applications to sentiment prediction, style analysis, and text summarization.

## 1 Introduction

The World Wide Web and other textual databases provide a convenient platform for exchanging opinions. Many documents, such as reviews and blogs, are written with the purpose of conveying a particular opinion or sentiment. Other documents may not be written with the purpose of conveying an opinion, but nevertheless they contain one. Opinions, or sentiments, may be considered in several ways, the simplest of which is varying from positive opinion, through neutral, to negative opinion.

Most of the research in information retrieval has focused on predicting the topic of a document, or its relevance with respect to a query. Predicting the document's sentiment would allow matching the sentiment, as well as the topic, with the user's interests. It would also assist in document summarization and visualization. Sentiment prediction was first formulated as a binary classification problem to answer questions such as: "What is the review's polarity, positive or negative?" Pang et al. [1] demonstrated the difficulties in sentiment prediction using solely the empirical rules (a subset of adjectives), which motivates the use of statistical learning techniques. The task was then refined to allow multiple sentiment levels, facilitating the use of standard text categorization techniques [2].

However, sentiment prediction is different from traditional text categorization: (1) in contrast to the categorical nature of topics, sentiments are ordinal variables; (2) several contradicting opinions might co-exist, which interact with each other to produce the global document sentiment; (3) context plays a vital role in determining the sentiment. Indeed, sentiment prediction is a much harder task than topic classification tasks such as Reuters or WebKB and current models achieve lower accuracy.

Rather than using a bag of words multiclass classifier, we model the sequential flow of sentiment throughout the document using a sequential conditional model. Furthermore, we treat the sentiment labels as ordinal variables by enforcing monotonicity constraints on the model's parameters.

## 2 Local and Global Sentiments

Previous research on sentiment prediction has generally focused on predicting the sentiment of the entire document. A commonly used application is the task of predicting the number of stars assigned

to a movie, based on a review text. Typically, the problem is considered as standard multiclass classification or regression using the bag of words representation.

In addition to the sentiment of the entire document, which we call global sentiment, we define the concept of local sentiment as the sentiment associated with a particular part of the text. It is reasonable to assume that the global sentiment of a document is a function of the local sentiment and that estimating the local sentiment is a key step in predicting the global sentiment. Moreover, the concept of local sentiment is useful in a wide range of text analysis applications including document summarization and visualization.

Formally, we view local sentiment as a function on the words in a document taking values in a finite partially ordered set, or a poset, $(O, \leq)$. To determine the local sentiment at a particular word, it is necessary to take context into account. For example, due to context the local sentiment at each of the following words `this is a horrible product` is low (in the sense of $(O, \leq)$). Since sentences are natural components for segmenting document semantics, we view local sentiment as a piecewise constant function on sentences. Occasionally we encounter a sentence that violates this rule and conveys opposing sentiments in two different parts. In this situation we break the sentence into two parts and consider them as two sentences. We therefore formalize the problem as predicting a sequence of sentiments $\mathbf{y} = (y_1, \ldots, y_n), y_i \in O$ based on a sequence of sentences $\mathbf{x} = (x_1, \ldots, x_n)$.

Modeling the local sentiment is challenging from several aspects. The sentence sequence $\mathbf{x}$ is discrete-time and high-dimensional categorical valued, and the sentiment sequence $\mathbf{y}$ is discrete-time and ordinal valued. Regression models can be applied locally but they ignore the statistical dependencies across the time domain. Popular sequence models such as HMM or CRF, on the other hand, typically assume that $\mathbf{y}$ is categorical valued. In this paper we demonstrate the prediction of local sentiment flow using an ordinal version of conditional random fields, and explore the relation between the local and global sentiment.

## 3   Isotonic Conditional Random Fields

Conditional random fields (CRF) [3] are parametric families of conditional distributions $p_\theta(\mathbf{y}|\mathbf{x})$ that correspond to undirected graphical models or Markov random fields

$$p_\theta(\mathbf{y}|\mathbf{x}) = \frac{p_\theta(\mathbf{y}, \mathbf{x})}{p_\theta(\mathbf{x})} = \frac{\prod_{c \in C} \phi_c(\mathbf{x}|_c, \mathbf{y}|_c)}{Z(\theta, \mathbf{x})} = \frac{\exp\left(\sum_{c \in C} \sum_k \theta_{c,k} f_{c,k}(\mathbf{x}|_c, \mathbf{y}|_c)\right)}{Z(\theta, \mathbf{x})} \quad \theta_{c,k} \in \mathbb{R} \quad (1)$$

where $C$ is the set of cliques in the graph and $\mathbf{x}|_c$ and $\mathbf{y}|_c$ are the restriction of $\mathbf{x}$ and $\mathbf{y}$ to variables representing nodes in $c \in C$. It is assumed above that the potentials $\phi_c$ are exponential functions of features modulated by decay parameters $\phi_c(\mathbf{x}|_c, \mathbf{y}|_c) = \exp(\sum_k \theta_{c,k} f_{c,k}(\mathbf{x}|_c, \mathbf{y}|_c))$.

CRF have been mostly applied to sequence annotation, where $\mathbf{x}$ is a sequence of words and $\mathbf{y}$ is a sequence of labels annotating the words, for example part-of-speech tags. The standard graphical structure in this case is a chain structure on $\mathbf{y}$ with noisy observations $\mathbf{x}$. In other words, the cliques are $C = \{\{y_{i-1}, y_i\}, \{y_i, x_i\} : i = 1, \ldots, n\}$ (see Figure 1 left) leading to the model

$$p_\theta(\mathbf{y}|\mathbf{x}) = \frac{1}{Z(\mathbf{x}, \theta)} \exp\left(\sum_i \sum_k \lambda_k f_k(y_{i-1}, y_i) + \sum_i \sum_k \mu_k g_k(y_i, x_i)\right) \qquad \theta = (\lambda, \mu). \quad (2)$$

In sequence annotation a standard choice for the feature functions is $f_{\langle \sigma, \tau \rangle}(y_{i-1}, y_i) = \delta_{y_{i-1}, \sigma} \delta_{y_i, \tau}$ and $g_{\langle \sigma, w \rangle}(y_i, x_i) = \delta_{y_i, \sigma} \delta_{x_i, w}$ (note that we index the feature functions using pairs rather than $k$ as in (2)). In our case, since $x_i$ are sentences we use instead the slightly modified feature functions $g_{\langle \sigma, w \rangle}(y_i, x_i) = 1$ if $y_i = \sigma, w \in x_i$ and 0 otherwise. Given a set of iid training samples the parameters are typically estimated by maximum likelihood or MAP using standard numerical techniques such as conjugate gradient or quasi-Newton.

Despite the great popularity of CRF in sequence labeling, they are not appropriate for ordinal data such as sentiments. The ordinal relation is ignored in (2), and in the case of limited training data the parameter estimates will possess high variance resulting in poor predictive power. We therefore enforce a set of monotonicity constraints on the parameters that are consistent with the ordinal structure and domain knowledge. The resulting model is a restricted subset of the CRF (2) and, in accordance with isotonic regression [4], is named isotonic CRF.

Since ordinal variables express a progression of some sort, it is natural to expect some of the binary features in (2) to correlate more strongly with some ordinal values than others. In such cases, we should expect the presence of such binary features to increase (or decrease) the conditional probability in a manner consistent with the ordinal relation. Since the parameters $\mu_{\langle\sigma,w\rangle}$ represent the effectiveness of the appearance of $w$ with respect to increasing the probability of $\sigma \in O$, they are natural candidates for monotonicity constraints. More specifically, for words $w \in \mathcal{M}_1$ that are identified as strongly associated with positive sentiment, we enforce

$$\sigma \leq \sigma' \quad \Longrightarrow \quad \mu_{\langle\sigma,w\rangle} \leq \mu_{\langle\sigma',w\rangle} \qquad \forall w \in \mathcal{M}_1. \tag{3}$$

Similarly, for words $w \in \mathcal{M}_2$ identified as strongly associated with negative sentiment, we enforce

$$\sigma \leq \sigma' \quad \Longrightarrow \quad \mu_{\langle\sigma,w\rangle} \geq \mu_{\langle\sigma',w\rangle} \qquad \forall w \in \mathcal{M}_2. \tag{4}$$

The motivation behind the above restriction is immediate for the non-conditional Markov random fields $p_\theta(\mathbf{x}) = Z^{-1} \exp(\sum \theta_i f_i(\mathbf{x}))$. Parameters $\theta_i$ are intimately tied to model probabilities through activation of the feature functions $f_i$. In the case of conditional random fields, things get more complicated due to the dependence of the normalization term on $\mathbf{x}$. The following propositions motivate the above parameter restriction for the case of linear structure CRF with binary features.

**Proposition 1.** *Let* $p(\mathbf{y}|\mathbf{x})$ *be a linear state-emission chain CRF with binary features* $f_{\langle\sigma,\tau\rangle}$, $g_{\langle\sigma,w\rangle}$ *as above, and* $\mathbf{x}$ *a sentence sequence for which* $v \notin x_j$. *Then, denoting* $\mathbf{x}' = (x_1, \ldots, x_{j-1}, x_j \cup \{v\}, x_{j+1}, \ldots, x_n)$, *we have*

$$\forall \mathbf{y} \qquad \frac{p(\mathbf{y}|\mathbf{x})}{p(\mathbf{y}|\mathbf{x}')} = E_{p(\mathbf{y}'|\mathbf{x})}\left(e^{\mu_{\langle y_j', v\rangle} - \mu_{\langle y_j, v\rangle}}\right).$$

*Proof.*

$$\frac{p(\mathbf{y}|\mathbf{x})}{p(\mathbf{y}|\mathbf{x}')} = \frac{Z(\mathbf{x}')}{Z(\mathbf{x})} \frac{e^{\left(\sum_i \sum_{\sigma,\tau} \lambda_{\langle\sigma,\tau\rangle} f_{\langle\sigma,\tau\rangle}(y_{i-1},y_i) + \sum_i \sum_{\sigma,w} \mu_{\langle\sigma,w\rangle} g_{\langle\sigma,w\rangle}(y_i, x_i)\right)}}{e^{\left(\sum_i \sum_{\sigma,\tau} \lambda_{\langle\sigma,\tau\rangle} f_{\langle\sigma,\tau\rangle}(y_{i-1},y_i) + \sum_i \sum_{\sigma,w} \mu_{\langle\sigma,w\rangle} g_{\langle\sigma,w\rangle}(y_i, x_i')\right)}} = \frac{Z(\mathbf{x}')}{Z(\mathbf{x})} e^{-\mu_{\langle y_j, v\rangle}}$$

$$= \frac{\sum_{\mathbf{y}'} e^{\left(\sum_i \sum_{\sigma,\tau} \lambda_{\langle\sigma,\tau\rangle} f_{\langle\sigma,\tau\rangle}(y_{i-1}',y_i') + \sum_i \sum_{\sigma,w} \mu_{\langle\sigma,w\rangle} g_{\langle\sigma,w\rangle}(y_i', x_i')\right)}}{\sum_{\mathbf{y}'} e^{\left(\sum_i \sum_{\sigma,\tau} \lambda_{\langle\sigma,\tau\rangle} f_{\langle\sigma,\tau\rangle}(y_{i-1}',y_i') + \sum_i \sum_{\sigma,w} \mu_{\langle\sigma,w\rangle} g_{\langle\sigma,w\rangle}(y_i', x_i)\right)}} e^{-\mu_{\langle y_j, v\rangle}}$$

$$= \frac{\sum_{r \in O} \alpha_r e^{\mu_{\langle r, v\rangle}}}{\sum_{r \in O} \alpha_r} e^{-\mu_{\langle y_j, v\rangle}} = \sum_{r \in O} \frac{\alpha_r}{\sum_{r' \in O} \alpha_{r'}} e^{\mu_{\langle r, v\rangle} - \mu_{\langle y_j, v\rangle}}$$

$$= \sum_{\mathbf{y}'} p(\mathbf{y}'|\mathbf{x}) e^{\mu_{\langle y_j', v\rangle} - \mu_{\langle y_j, v\rangle}}$$

where

$$\alpha_r = \sum_{\mathbf{y}': y_j' = r} \exp\left(\sum_i \sum_{\sigma,\tau} \lambda_{\langle\sigma,\tau\rangle} f_{\langle\sigma,\tau\rangle}(y_{i-1}', y_i') + \sum_i \sum_{\sigma,w} \mu_{\langle\sigma,w\rangle} g_{\langle\sigma,w\rangle}(y_i', x_i)\right).$$

$\square$

Note that the specific linear CRF structure (Figure 1, left) and binary features are essential for the above result. Proposition 1 connects the probability ratio $\frac{p(\mathbf{y}|\mathbf{x})}{p(\mathbf{y}|\mathbf{x}')}$ to the model parameters in a relatively simple manner. Together with Proposition 2 below, it motivates the ordering of $\{\mu_{\langle r,v\rangle} : r \in O\}$ determined by the restrictions (3)-(4) in terms of the ordering of probability ratios of transformed sequences.

**Proposition 2.** *Let* $p(\mathbf{y}|\mathbf{x}), \mathbf{x}, \mathbf{x}'$ *be as in Proposition 1. For all label sequences* $\mathbf{s}, \mathbf{t}$, *we have*

$$\mu_{\langle t_j, v\rangle} \geq \mu_{\langle s_j, v\rangle} \qquad \Longrightarrow \qquad \frac{p(\mathbf{s}|\mathbf{x})}{p(\mathbf{s}|\mathbf{x}')} \geq \frac{p(\mathbf{t}|\mathbf{x})}{p(\mathbf{t}|\mathbf{x}')}. \tag{5}$$

*Proof.* Since $\mu_{\langle t_j, v\rangle} \geq \mu_{\langle s_j, v\rangle}$ we have that $e^{z - \mu_{\langle s_j, v\rangle}} - e^{z - \mu_{\langle t_j, v\rangle}} \geq 0$ for all $z$ and

$$E_{p(\mathbf{y}'|\mathbf{x})}\left(e^{\mu_{\langle y_j', v\rangle} - \mu_{\langle s_j, v\rangle}} - e^{\mu_{\langle y_j', v\rangle} - \mu_{\langle t_j, v\rangle}}\right) \geq 0.$$

By Proposition 1 the above expectation is $\frac{p(\mathbf{s}|\mathbf{x})}{p(\mathbf{s}|\mathbf{x}')} - \frac{p(\mathbf{t}|\mathbf{x})}{p(\mathbf{t}|\mathbf{x}')}$ and Equation (5) follows. $\square$

The restriction (3) may thus be interpreted as ensuring that adding a word $w \in \mathcal{M}_1$ to transform $\mathbf{x} \mapsto \mathbf{x}'$ will increase labeling probabilities associated with $\sigma$ no less than with $\sigma'$ if $\sigma' \leq \sigma$. Similarly, the restriction (4) may be interpreted in the opposite way. If these assumptions are correct, it is clear that they will lead to more accurate parameter estimates and better prediction accuracy. However, even if assumptions (3)-(4) are incorrect, enforcing them may improve prediction by trading off increased bias with lower variance.

Conceptually, the parameter estimates for isotonic CRF may be found by maximizing the likelihood or posterior subject to the monotonicity constraints (3)-(4). Since such a maximization is relatively difficult for large dimensionality, we propose a re-parameterization that leads to a much simpler optimization problem. The re-parameterization, in the case of a fully ordered set, is relatively straightforward. In the more general case of a partially ordered set we need the mechanism of Möbius inversions on finite partially ordered sets.

We introduce a new set of features $\{g^*_{\langle \sigma, w \rangle} : \sigma \in O\}$ for $w \in \mathcal{M}_1 \cup \mathcal{M}_2$ defined as

$$g^*_{\langle \sigma, w \rangle}(y_i, x_i) = \sum_{\tau : \tau \geq \sigma} g_{\langle \tau, w \rangle}(y_i, x_i) \qquad w \in \mathcal{M}_1 \cup \mathcal{M}_2$$

and a new set of corresponding parameters $\{\mu^*_{\langle \sigma, w \rangle} : \sigma \in O\}$. If $(O, \leq)$ is fully ordered, $\mu^*_{\langle \sigma, w \rangle} = \mu_{\langle \sigma, w \rangle} - \mu_{\langle \sigma', w \rangle}$, where $\sigma'$ is the largest element smaller than $\sigma$, or 0 if $\sigma = \min(O)$. In the more general case, $\mu^*_{\langle \sigma, w \rangle}$ is the convolution of $\mu_{\langle \sigma, w \rangle}$ with the Möbius function of the poset $(O, \leq)$ (see [5] for more details). By the Möbius inversion theorem [5] we have that $\mu^*_{\langle \sigma, w \rangle}$ satisfy

$$\mu_{\langle \sigma, w \rangle} = \sum_{\tau : \tau \leq \sigma} \mu^*_{\langle \tau, w \rangle} \qquad w \in \mathcal{M}_1 \cup \mathcal{M}_2 \qquad (6)$$

and that $\sum_\tau \mu_{\langle \tau, w \rangle} g_{\langle \tau, w \rangle} = \sum_\tau \mu^*_{\langle \tau, w \rangle} g^*_{\langle \tau, w \rangle}$ leading to the re-parameterization of isotonic CRF

$$p(\mathbf{y}|\mathbf{x}) = \frac{1}{Z(\mathbf{x})} \exp \left( \sum_i \sum_{\sigma, \tau} \lambda_{\langle \sigma, \tau \rangle} f_{\langle \sigma, \tau \rangle}(y_{i-1}, y_i) + \sum_i \sum_{w \notin \mathcal{M}_1 \cup \mathcal{M}_2} \sum_\sigma \mu_{\langle \sigma, w \rangle} g_{\langle \sigma, w \rangle}(y_i, x_i) \right.$$

$$\left. + \sum_i \sum_{w \in \mathcal{M}_1 \cup \mathcal{M}_2} \sum_\sigma \mu^*_{\langle \sigma, w \rangle} g^*_{\langle \sigma, w \rangle}(y_i, x_i) \right)$$

with $\mu^*_{\langle \sigma, w \rangle} \geq 0, w \in \mathcal{M}_1$ and $\mu^*_{\langle \sigma, w \rangle} \leq 0, w \in \mathcal{M}_2$ for all $\sigma > \min(O)$. The re-parameterized model has the benefit of simple constraints and its maximum likelihood estimates can be obtained by a trivial adaptation of conjugate gradient or quasi-Newton methods.

## 3.1 Author Dependent Models

Thus far, we have ignored the dependency of the labeling model $p(\mathbf{y}|\mathbf{x})$ on the author, denoted here by the variable $a$. We now turn to account for different sentiment-authoring styles by incorporating this variable into the model. The word emissions $y_i \rightarrow x_i$ in the CRF structure are not expected to vary much across different authors. The sentiment transitions $y_{i-1} \rightarrow y_i$, on the other hand, typically vary across different authors as a consequence of their individual styles. For example, the review of an author who sticks to a list of self-ranked evaluation criteria is prone to strong sentiment variations. In contrast, the review of an author who likes to enumerate pros before he gets to cons (or vice versa) is likely to exhibit more local homogeneity in sentiment.

Accounting for author-specific sentiment transition style leads to the graphical model in Figure 1 right. The corresponding author-dependent CRF model

$$p(\mathbf{y}|\mathbf{x}, a) = \frac{1}{Z(\mathbf{x}, a)} e^{\left( \sum_{i, a'} \sum_{\sigma, \tau} \left( \lambda_{\langle \sigma, \tau \rangle} + \lambda_{\langle \sigma, \tau, a' \rangle} \right) f_{\langle \sigma, \tau, a' \rangle}(y_{i-1}, y_i, a) + \sum_i \sum_{\sigma, w} \mu_{\langle \sigma, w \rangle} g_{\langle \sigma, w \rangle}(y_i, x_i) \right)}$$

uses features $f_{\langle \sigma, \tau, a' \rangle}(y_{i-1}, y_i, a) = f_{\langle \sigma, \tau \rangle}(y_{i-1}, y_i) \delta_{a, a'}$ and transition parameters that are author-dependent $\lambda_{\langle \sigma, \tau, a \rangle}$ as well as author-independent $\lambda_{\langle \sigma, \tau \rangle}$. Setting $\lambda_{\langle \sigma, \tau, a \rangle} = 0$ reduces the model to the standard CRF model. The author-independent parameters $\lambda_{\langle \sigma, \tau \rangle}$ allow parameter sharing across multiple authors in case the training data is too scarce for proper estimation of $\lambda_{\langle \sigma, \tau, a \rangle}$. For simplicity, the above ideas are described in the context of non-isotonic CRF. However, it is straightforward to combine author-specific models with isotonic restrictions. Experiments demonstrating author-specific isotonic models are described in Section 4.3.

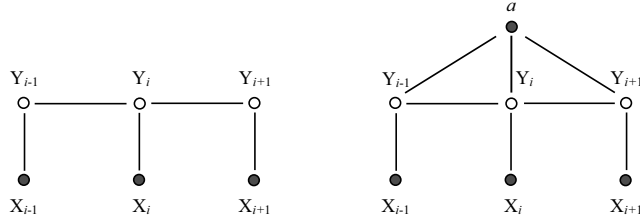

Figure 1: Graphical models corresponding to CRF (left) and author-dependent CRF (right).

## 3.2 Sentiment Flows as Smooth Curves

The sentence-based definition of sentiment flow is problematic when we want to fit a model (for example to predict global sentiment) that uses sentiment flows from multiple documents. Different documents have different number of sentences and it is not clear how to compare them or how to build a model from a collection of discrete flows of different lengths. We therefore convert the sentence-based flow to a smooth length-normalized flow that can meaningfully relate to other flows.

We assume from now on that the ordinal set $O$ is realized as a subset of $\mathbb{R}$ and that its ordering coincides with the standard ordering on $\mathbb{R}$. In order to account for different lengths, we consider the sentiment flow as a function $h : [0, 1] \rightarrow O \subset \mathbb{R}$ that is piecewise constant on the intervals $[0, l), [l, 2l), \dots, [(k - 1)l, 1]$ where $k$ is the number of sentences in the document and $l = 1/k$. Each of the intervals represents a sentence and the function value on it is its sentiment.

To create a more robust representation we smooth out the discontinuous function by convolving it with a smoothing kernel. The resulting sentiment flow is a smooth curve $f : [0, 1] \rightarrow \mathbb{R}$ that can be easily related or compared to similar sentiment flows of other documents (see Figure 3 for an example). We can then define natural distances between two flows, for example the $L_p$ distance

$$d_p(f_1, f_2) = \left( \int_0^1 |f_1(r) - f_2(r)|^p \, dr \right)^{1/p} \tag{7}$$

for use in a $k$-nearest neighbor model for relating the local sentiment flow to the global sentiment.

## 4 Experiments

To examine the ideas proposed in this paper we implemented isotonic CRF, and the normalization and smoothing procedure, and experimented with a small dataset of 249 movie reviews, randomly selected from the Cornell sentence polarity dataset v1.0[1], all written by the same author. The code for isotonic CRF is a modified version of the quasi-Newton implementation in the Mallet toolkit. In order to check the accuracy and benefit of the local sentiment predictor, we hand-labeled the local sentiments of each of these reviews. We assigned for each sentence one of the following values in $O \subset \mathbb{R}$: 2 (highly praised), 1 (something good), 0 (objective description), $-1$ (something that needs improvement) and $-2$ (strong aversion).

### 4.1 Sentence Level Prediction

To evaluate the prediction quality of the local sentiment we compared the performance of naive Bayes, SVM (using the default parameters of SVM$^{light}$), CRF and isotonic CRF. Figure 2 displays the testing accuracy and distance of predicting the sentiment of sentences as a function of the training data size averaged over 20 cross-validation train-test split.

The dataset presents one particular difficulty where more than 75% of the sentences are labeled objective (or 0). As a result, the prediction accuracy for objective sentences is over-emphasized. To correct for this fact, we report our test-set performance over a balanced (equal number of sentences for different labels) sample of labeled sentences. Note that since there are 5 labels, random guessing yields a baseline of 0.2 accuracy and guessing 0 always yields a baseline of 1.2 distance.

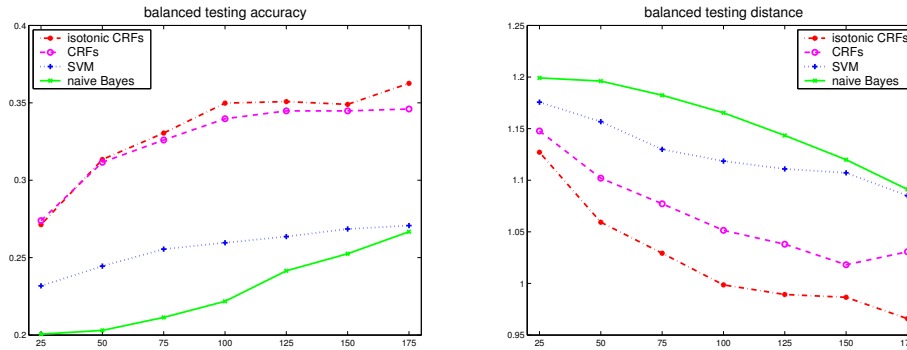

Figure 2: Local sentiment prediction: balanced test results for naive Bayes, SVM, CRF and iso-CRF.

As described in Section 3, for isotonic CRF, we obtained 300 words to enforce monotonicity constraints. The 150 words that achieved the highest correlation with the sentiment were chosen for positivity constraints. Similarly, the 150 words that achieved the lowest correlation were chosen for negativity constraints. Table 1 displays the top 15 words of the two lists.

| great | superb | memorable | enjoyable | mood |
|---|---|---|---|---|
| perfection | outstanding | performance | enjoyed | certain |
| considerable | wonderfully | worth | beautifully | delightfully |
| too | didnt | just | failed | unnecessary |
| couldnt | i | no | satire | contrived |
| wasnt | uninspired | lacked | boring | tended |

Table 1: Lists of 15 words with the largest positive (top) and negative (bottom) correlations.

The results in Figure 2 indicate that by incorporating the sequential information, the two versions of CRF perform consistently better than SVM and naive Bayes. The advantage of setting the monotonicity constraints in CRF is elucidated by the average absolute distance performance criterion (Figure 2, right). This criterion is based on the observation that in sentiment prediction, the cost of misprediction is influenced by the ordinal relation on the labels, rather than the 0-1 error rate.

## 4.2 Global Sentiment Prediction

We also evaluated the contribution of the local sentiment analysis in helping to predict the global sentiment of documents. We compared a nearest neighbor classifier for the global sentiment, where the representation varied from bag of words to smoothed length-normalized local sentiment representation (with and without objective sentences). The smoothing kernel was a bounded Gaussian density (truncated and renormalized) with $\sigma^2 = 0.2$. Figure 3 displays discrete and smoothed local sentiment labels, and the smoothed sentiment flow predicted by isotonic CRF.

Figure 4 and Table 2 display test-set accuracy of global sentiments as a function of the train set size. The distance in the nearest neighbor classifier was either $L_1$ or $L_2$ for the bag of words representation or their continuous version (7) for the smoothed sentiment curve representation. The results indicate that the classification performance of the local sentiment representation is better than the bag of words representation. In accordance with the conclusion of [6], removing objective sentences (that correspond to sentiment 0) increased the local sentiment analysis performance by 20.7%. We can thus conclude that for the purpose of global sentiment prediction, local sentiment flow of the non-objective sentences holds most of the relevant information. Performing local sentiment analysis on non-objective sentences improves performance as the model estimates possess lower variance.

## 4.3 Measuring the rate of sentiment change

We examine the rate of sentiment change as a characterization of the author's writing style using the isotonic author-dependent model of Section 3.1. We assume that the CRF process is a discrete

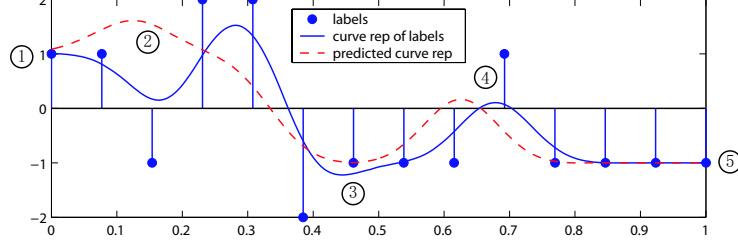

Figure 3: Sentiment flow and its smoothed curve representation. The blue circles indicate the labeled sentiment of each sentence. The blue solid curve and red dashed curve are smoothed representations of the labeled and predicted sentiment flows. Only non-objective labels are kept in generating the two curves. The numberings correspond to sentences displayed in Section 4.4.

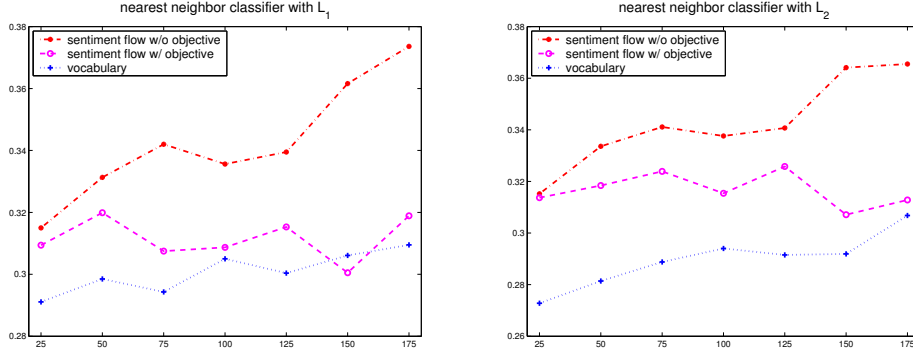

Figure 4: Accuracy of global sentiment prediction (4-class labeling) as a function of train set size.

sampling of a corresponding continuous time Markov jump process. A consequence of this assumption is that the time $T$ the author stays in sentiment $\sigma$ before leaving is modeled by the exponential distribution $p_\sigma(T > t) = e^{-q_\sigma(t-1)}, t > 1$. Here, we assume $T > 1$ and $q_\sigma$ is interpreted as the rate of change of the sentiment $\sigma \in O$: the larger the value, the more likely the author will switch to other sentiments in the near future.

To estimate the rate of change $q_\sigma$ of an author we need to compute $p_\sigma(T > t)$ based on the marginal probabilities $p(\mathbf{s}|a)$ of sentiment sequences $\mathbf{s}$ of length $l$. The probability $p(\mathbf{s}|a)$ may be approximated by

$$p(\mathbf{s}|a) = \sum_{\mathbf{x}} p(\mathbf{x}|a)p(\mathbf{s}|\mathbf{x}, a) \tag{8}$$

$$\approx \sum_{\mathbf{x}} \frac{\tilde{p}'(\mathbf{x}|a)}{n - l + 1} \left( \sum_i \frac{\alpha_i(s_1|\mathbf{x}, a) \prod_{j=i+1}^{i+(l-1)} M_j(s_{j-i}, s_{j-i+1}|\mathbf{x}, a)\beta_{i+(l-1)}(s_l|\mathbf{x}, a)}{Z(\mathbf{x}, a)} \right)$$

where $\tilde{p}'$ is the empirical probability function $\tilde{p}'(\mathbf{x}|a) = \frac{1}{|C|} \sum_{\mathbf{x}' \in C} \delta_{\mathbf{x},\mathbf{x}'}$ for the set $C$ of documents written by author $a$ of length no less than $l$. $\alpha, M, \beta$ are the forward, transition and backward probabilities analogous to the dynamic programming method in [3].

Using the model $p(\mathbf{s}|a)$ we can compute $p_\sigma(T > t)$ for different authors at integer values of $t$ which would lead to the quantity $q_\sigma$ associated with each author. However, since (8) is based on an approximation, the calculated values of $p_\sigma(T > t)$ will be noisy resulting in slightly different values of $q_\sigma$ for different time points $t$ and cross validation iterations. A linear regression fit for $q_\sigma$ based on the approximated values of $p_\sigma(T > t)$ for two authors using 10-fold cross validation is displayed in Figure 5. The data was the 249 movie reviews from the previous experiments written by one author, and additional 201 movie reviews from a second author. Interestingly, the author associated with the red dashed line has a consistent lower $q_\sigma$ value in all those figures, and thus is considered as more "static" and less prone to quick sentiment variations.

| | $L_1$ | | $L_2$ | |
|---|---|---|---|---|
| vocabulary | 0.3095 | | 0.3068 | |
| sentiment flow with objective sentences | 0.3189 | 3.0% | 0.3128 | 1.95% |
| sentiment flow without objective sentences | 0.3736 | 20.7% | 0.3655 | 19.1% |

Table 2: Accuracy results and relative improvement when training size equals 175.

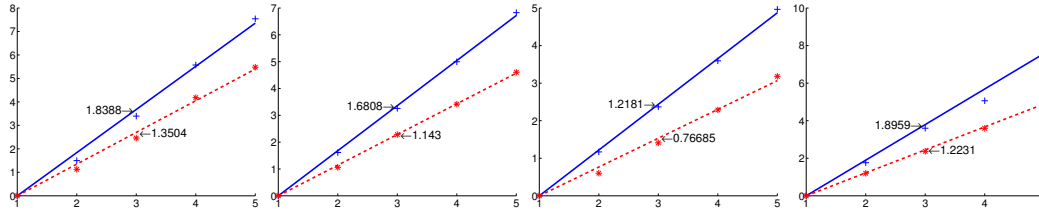

Figure 5: Linear regression fit for $q_\sigma$, $\sigma = 2, 1, -1, -2$ (left to right) based on approximated values of $p_\sigma(T > t)$ for two different authors. X-axis: time $t$; Y-axis: negative log-probability of $T > t$.

## 4.4 Text Summarization

We demonstrate the potential usage of sentiment flow for text summarization with a very simple example. The text below shows the result of summarizing the movie review in Figure 3 by keeping only sentences associated with the start, the end, the top, and the bottom of the predicted sentiment curve. The number before each sentence relates to the circled number in Figure 3.

1 What makes this film mesmerizing, is not the plot, but the virtuoso performance of Lucy Berliner (Ally Sheedy), as a wily photographer, retired from her professional duties for the last ten years and living with a has-been German actress, Greta (Clarkson). 2 The less interesting story line involves the ambitions of an attractive, baby-faced assistant editor at the magazine, Syd (Radha Mitchell), who lives with a boyfriend (Mann) in an emotionally chilling relationship. 3 We just lost interest in the characters, the film began to look like a commercial for a magazine that wouldn't stop and get to the main article. 4 Which left the film only somewhat satisfying; it did create a proper atmosphere for us to view these lost characters, and it did have something to say about how their lives are being emotionally torn apart. 5 It would have been wiser to develop more depth for the main characters and show them to be more than the superficial beings they seemed to be on screen.

Alternative schemes for extracting specific sentences may be used to achieve different effects, depending on the needs of the user. We plan to experiment further in this area by combining local sentiment flow and standard summarization techniques.

## 5 Discussion

In this paper, we address the prediction and application of the local sentiment flow concept. As existing models are inadequate for a variety of reasons, we introduce the isotonic CRF model that is suited to predict the local sentiment flow. This model achieves better performance than the standard CRF as well as non-sequential models such as SVM. We also demonstrate the usefulness of the local sentiment representation for global sentiment prediction, style analysis and text summarization.

## Footnotes

[1]Available at http://www.cs.cornell.edu/People/pabo/movie-review-data

## References

[1] B. Pang, L. Lee, and S. Vaithyanathan. Thumbs up? sentiment classification using machine learning techniques. In *Proceedings of EMNLP-02*.

[2] B. Pang and L. Lee. Seeing stars: Exploiting class relationships for sentiment categorization with respect to rating scales. In *Proceedings of ACL-05*.

[3] J. Lafferty, F. Pereira, and A. McCallum. Conditional random fields: Probabilistic models for segmenting and labeling sequence data. In *International Conference on Machine Learning*, 2001.

[4] R. E. Barlow, D.J. Bartholomew, J. M. Bremner, and H. D. Brunk. *Statistical inference under order restrictions; the theory and application of isotonic regression*. Wiley, 1972.

[5] R. P. Stanley. *Enumerative Combinatorics*. Wadsworth & Brooks/Cole Mathematics Series, 1986.

[6] B. Pang and L. Lee. A sentimental education: Sentiment analysis using subjectivity summarization based on minimum cuts. In *Proceedings of ACL-04*.
